# The Price of Bandit Information for Online Optimization

**Varsha Dani**
Department of Computer Science
University of Chicago
Chicago, IL 60637
varsha@cs.uchicago.edu

**Thomas P. Hayes**
Toyota Technological Institute
Chicago, IL 60637
hayest@tti-c.org

**Sham M. Kakade**
Toyota Technological Institute
Chicago, IL 60637
sham@tti-c.org

## Abstract

In the online linear optimization problem, a learner must choose, in each round, a decision from a set $D \subset \mathbb{R}^n$ in order to minimize an (unknown and changing) linear cost function. We present sharp rates of convergence (with respect to additive regret) for both the full information setting (where the cost function is revealed at the end of each round) and the bandit setting (where only the scalar cost incurred is revealed). In particular, this paper is concerned with the *price of bandit information*, by which we mean the ratio of the best achievable regret in the bandit setting to that in the full-information setting. For the full information case, the upper bound on the regret is $O^*(\sqrt{nT})$, where $n$ is the ambient dimension and $T$ is the time horizon. For the bandit case, we present an algorithm which achieves $O^*(n^{3/2}\sqrt{T})$ regret — all previous (nontrivial) bounds here were $O(\mathrm{poly}(n)T^{2/3})$ or worse. It is striking that the convergence rate for the bandit setting is only a factor of $n$ worse than in the full information case — in stark contrast to the $K$-arm bandit setting, where the gap in the dependence on $K$ is exponential ($\sqrt{TK}$ vs. $\sqrt{T \log K}$). We also present lower bounds showing that this gap is at least $\sqrt{n}$, which we conjecture to be the correct order. The bandit algorithm we present can be implemented efficiently in special cases of particular interest, such as path planning and Markov Decision Problems.

## 1 Introduction

In the online linear optimization problem (as in Kalai and Vempala [2005]), at each timestep the learner chooses a decision $x_t$ from a decision space $D \subset \mathbb{R}^n$ and incurs a cost $L_t \cdot x_t$, where the loss vector $L_t$ is in $\mathbb{R}^n$. This paper considers the case where the sequence of loss vectors $L_1, \ldots, L_T$ is arbitrary — that is, no statistical assumptions are made about the data generation process. The goal of the learner is to minimize her regret, the difference between the incurred loss on the sequence and the loss of the best single decision in hindsight. After playing $x_t$ at time $t$, the two most natural sources of feedback that the learner receives are either complete information of the loss vector $L_t$ (referred to as the full information case) or only the scalar feedback of the incurred loss $L_t \cdot x_t$ (referred to as the partial feedback or "bandit" case).

The online linear optimization problem has been receiving increasing attention as a paradigm for structured decision making in dynamic environments, with potential applications to network routing,

| | K-Arm | | Linear Optimization | | | |
|---|---|---|---|---|---|---|
| | Full | Partial | Full | Partial | | |
| | | | | I.I.D. | Expectation | High Probability |
| Lower Bound | $\sqrt{T \ln K}$ | $\sqrt{TK}$ | $\mathbf{\sqrt{nT}}$ | $\mathbf{n\sqrt{T}}$ | $\mathbf{n\sqrt{T}}$ | $\mathbf{n\sqrt{T}}$ |
| Upper Bound | $\sqrt{T \ln K}$ | $\sqrt{TK}$ | $\mathbf{\sqrt{nT}}$ | $n\sqrt{T}$ | $\mathbf{n^{3/2}\sqrt{T}}$ | $n^{3/2}\sqrt{T}$ |
| Efficient Algo | N/A | N/A | Sometimes | Yes | Sometimes | ? |

Table 1: Summary of Regret Bounds: Only the leading dependency in terms of $n$ and $T$ are shown (so some log factors are dropped). The results in bold are provided in this paper. The results for the $K$-arm case are from Freund and Schapire [1997], Auer et al. [1998]. The i.i.d. column is the stochastic setting (where the loss vectors are drawn from some fixed underlying distribution) and the result are from Dani et al. [2008]. The expectation column refers to the expected regret for an arbitrary sequence of loss vectors (considered in this paper). The high probability column follows from a forthcoming paper Bartlett et al. [2007]; these results also hold in the adaptive adversary setting, where the loss vectors could change in response to the learner's previous decisions. The Efficient Algo row refers to whether or not there is an efficient implementation — "yes" means there is a polytime algorithm (for the stated upper bound) which only uses access to a certain optimization oracle (as in Kalai and Vempala [2005]) and "sometimes" means only in special cases (such as Path Planning) can the algorithm be implemented efficiently. See text for further details.

path planning, job scheduling, etc. This paper focuses on the fundamental regrets achievable for the online linear optimization problem in both the full and partial information feedback settings, as functions of both the dimensionality $n$ and the time horizon $T$. In particular, this paper is concerned with what might be termed *the price of bandit information* — how much worse the regret is in the partial information case as compared to the full information case.

In the $K$-arm case (where $D$ is the set of $K$ choices), much work has gone into obtaining sharp regret bounds. These results are summarized in the left two columns in Table 1. For the full information case, the exponential weights algorithm, **Hedge**, of Freund and Schapire [1997] provides the regret listed. For the partial information case, there is a long history of sharp regret bounds in various settings (particularly in statistical settings where i.i.d assumptions are made), dating back to Robbins [1952]. In the (non-statistical) adversarial case, the algorithm of Auer et al. [1998] provides the regret listed in Table 1 for the partial information setting. This case has a convergence rate that is exponentially worse than the full information case (as a function of $K$).

There are a number of issues that we must address in obtaining sharp convergence for the online linear optimization problem. The first issue to address is in understanding what are the natural quantities to state upper and lower bounds in terms of. It is natural to consider the case where the loss is uniformly bounded (say in $[0, 1]$). Clearly, the dimensionality $n$ and the time horizon $T$ are fundamental quantities. For the full information case, all previous bounds (see, e.g., Kalai and Vempala [2005]) also have dependencies on the diameter of the decision and cost spaces. It turns out that these are extraneous quantities — with the bounded loss assumption, one need not explicitly consider diameters of the decision and cost spaces. Hence, even in the full information case, to obtain a sharp upper bound we need a new argument to get an upper bound that is stated only in terms of $n$ and $T$ (and we do this via a relatively straightforward appeal to **Hedge**).

The second (and more technically demanding) issue is to obtain a sharp bound for the partial information case. Here, for the $K$-arm bandit case, the regret is $O^*(\sqrt{KT})$. Trivially, we can appeal to this result in the linear optimization case to obtain a $\sqrt{|D|T}$ regret by setting $K$ to be the size of $D$. However, the regret could have a very poor $n$ dependence, as $|D|$ could be exponential in $n$ (or worse). In contrast, note that in the full information case, we could appeal to the $K$-arm case to obtain $O(\sqrt{T \log |D|})$ regret, which in many cases is acceptable (such as when $D$ is exponential in $n$). The primary motivation for different algorithms in the full information case (e.g. Kalai and Vempala [2005]) was for computational reasons. In contrast, in the partial information case, we seek a new algorithm in order to just obtain a sharper convergence rate (of course, we are still also interested in efficient implementations). The goal here is provide a regret that is $O^*(\mathrm{poly}(n)\sqrt{T})$.

In fact, the partial information case (for linear optimization) has been receiving increasing interest in the literature [Awerbuch and Kleinberg, 2004, McMahan and Blum, 2004, Dani and Hayes, 2006]. Here, all regrets provided are $O(\mathrm{poly}(n)T^{2/3})$ or worse. We should note that some of the results here [Awerbuch and Kleinberg, 2004, Dani and Hayes, 2006] are stated in terms of only $n$

and $T$ (without referring to the diameters of various spaces). There is only one (non-trivial) special case [Gyorgy et al., 2007] in the literature where an $O^*(\text{poly}(n)\sqrt{T})$ regret has been established, and this case assumes significantly more feedback than in the partial information case — their result is for Path Planning (where $D$ is the set of paths on a graph and $n$ is the number of edges) and the feedback model assumes that learner receives the weight along *each* edge that is traversed (significantly more information than the just the scalar loss). The current paper provides the first $O^*(\text{poly}(n)\sqrt{T})$ regret for the general online linear optimization problem with scalar feedback — in particular, our algorithm has an expected regret that is $O^*(n^{3/2}\sqrt{T})$.

The final issue to address here is lower bounds, which are not extant in the literature. This paper provides lower bounds for both the full and partial information case. We believe these lower bounds are tight, up to log factors.

We have attempted to summarize the extant results in the literature (along with the results in this paper) in Table 1. We believe that we have a near complete picture of the achievable rates. One striking result is that the price of bandit information is relatively small — the upper bound is only a factor of $n$ worse than in the full information case. In fact, the lower bounds suggest the partial feedback case is only worse by a factor of $\sqrt{n}$. Contrast this to the $K$ arm case, where the full information case does exponentially better as a function of $K$.

As we believe that the lower bounds are sharp, we conjecture that the price of bandit information is only $\sqrt{n}$. Part of our reasoning is due to our previous result [Dani et al., 2008] in the i.i.d. case (where the linear loss functions are sampled from a fixed, time invariant distribution) — there, we provided an upper bound on the regret of only $O^*(n\sqrt{T})$. That bound was achieved by a *deterministic* algorithm which was a generalization of the celebrated algorithm of Lai and Robbins. [1985] for the $K$-arm case (in the i.i.d. setting).

Finally, we should note that this paper primarily focuses on the achievable regrets, not on efficient implementations. In much of the previous work in the literature (for both the full and partial information case), the algorithms can be implemented efficiently provided access to a certain optimization oracle. We are not certain whether our algorithms can be implemented efficiently, in general, with only this oracle access. However, as our algorithms use the **Hedge** algorithm of Freund and Schapire [1997], for certain important applications, efficient implementations *do* exist, based on dynamic programming. Examples include problems such as Path Planning (for instance, in routing network traffic), and also Markov Decision Problems, one of the fundamental models for long-term planning in AI. This idea has been developed by Takimoto and Warmuth [2003] and also applied by Gyorgy et al. [2007] (mentioned earlier) for Path Planning — the extension to Markov Decision Problems is relatively straightforward (based on dynamic programming).

The paper is organized as follows. In Section 2, we give a formal description of the problem. Then in Section 3 we present upper bounds for both the full information and bandit settings. Finally, in Section 4 we present lower bounds for both settings. All results in this paper are summarized in Table 1 (along with previous work).

## 2 Preliminaries

Let $D \subset \mathbb{R}^n$ denote the decision space. The learner plays the following $T$-round game against an oblivious adversary. First, the adversary chooses a sequence $L_1, \ldots, L_T$ of loss vectors in $\mathbb{R}^n$. We assume that the loss vectors are *admissible*, meaning they satisfy the boundedness property that for each $t$ and for all $x \in D$, $0 \le L_t \cdot x = L_t^\dagger x \le 1$. On each round $t$, the learner must choose a decision $x_t$ in $D$, which results in a loss of $\ell_t = L_t^\dagger x_t$. Throughout the paper we represent $x \in D$ and $L_t$ as column vectors and use $v^\dagger$ to denote the transpose of a column vector $v$. In the full information case, $L_t$ is revealed to the learner after time $t$. In the partial information case, only the incurred loss $\ell_t$ (and not the vector $L_t$) is revealed.

If $x_1, \ldots, x_T$ are the decisions the learner makes in the game, then the total loss is $\sum_{t=1}^T L_t^\dagger x_t$. The *cumulative regret* is defined by

$$R = \sum_{t=1}^T L_t^\dagger x_t - \min_{x \in D}\left(\sum_{t=1}^T L_t^\dagger x\right)$$

In other words, the learner's loss is compared to the loss of the best single decision in hindsight. The goal of the learner is to make a sequence of decisions that guarantees low regret. For the partial information case, our upper bounds on the regret are only statements that hold in expectation (with respect to the learner's randomness). The lower bounds provided hold with high probability.

This paper also assumes the learner has access to a *barycentric spanner* (as defined by Awerbuch and Kleinberg [2004]) of the decision region — such a spanner is useful for exploration. This is a subset of $n$ linearly independent vectors of the decision space, such that every vector in the decision space can be expressed as a linear combination of elements of the spanner with coefficients in $[-1, 1]$. Awerbuch and Kleinberg [2004] showed that any full rank compact set in $R^n$ has a barycentric spanner. Furthermore, an *almost* barycentric spanner (where the coefficients are in $[-2, 2]$) can be found efficiently (with certain oracle access). In view of these remarks, we assume without loss of generality, that $D$ contains the standard basis vectors $\vec{e}_1 \ldots \vec{e}_n$ and that $D \subset [-1, 1]^n$. We refer to the set $\{\vec{e}_1 \ldots \vec{e}_n\}$ as the *spanner*. Note that with this assumption, $\|x\|_2 \leq \sqrt{n}$ for all $x \in D$.

## 3 Upper Bounds

The decision set $D$ may be potentially large or even uncountably infinite. However, for the purposes of designing algorithms with sharp regret bounds, the following lemma shows that we need only concern ourselves with finite decision sets — the lemma shows that any decision set may be approximated to sufficiently high accuracy by a suitably small set (which is a $1/\sqrt{T}$-net for $D$).

**Lemma 3.1.** *Let $D \subset [-1, 1]^n$ be an arbitrary decision set. Then there is a set $\tilde{D} \subset D$ of size at most $(4nT)^{n/2}$ such that for every sequence of admissible loss vectors, the optimal loss for $\tilde{D}$ is within an additive $\sqrt{nT}$ of the optimal loss for $D$.*

*Proof sketch.* For each $x \in D$ suppose we truncate each coordinate of $x$ to only the first $\frac{1}{2}\log(nT)$ bits. Now from all $x \in D$ which result in the same truncated representation, we select a single representative to be included in $\tilde{D}$ This results in a set $\tilde{D}$ of size at most $(4nT)^{n/2}$ which is a $1/\sqrt{T}$-net for $D$. That is, every $x \in D$ is at distance at most $1/\sqrt{T}$ from its nearest neighbor in $\tilde{D}$. Since an admissible loss vector has norm at most $\sqrt{n}$, summing over the $T$ rounds of the game, we see that the optimal loss for $\tilde{D}$ is within an additive $\sqrt{nT}$ of the optimal loss for $D$. □

For implementation purposes, it may be impractical to store the decision set (or the covering net of the decision set) explicitly as a list of points. However, our algorithms only require the ability to sample from a specific distribution over the decision set. Furthermore, in many cases of interest the full decision set is finite and exponential in $n$, so we can directly work with $D$ (rather than a cover of $D$). As discussed in the Introduction, in many important cases of interest this can actually be accomplished using time and space which are only logarithmic in $|D|$ — this is due to that **Hedge** can be implemented efficiently for these special cases.

### 3.1 With Full Information

In the full information setting, the algorithm **Hedge** of Freund and Schapire [1997] guarantees a regret at most of $O(\sqrt{T \log |D|})$. Since we may modify $D$ so that $\log |D|$ is $O(n \log n \log T)$, this gives us regret $O^*(\sqrt{nT})$. Note that we are only concerned with the regret here. **Hedge** may in general be quite inefficient to implement. However, in many special cases of interest, efficient implementations *are* in fact possible, as discussed in the Introduction.

We also note that under the relatively minor assumption of the existence of an oracle for offline optimization, the algorithm of Kalai and Vempala [2005] *is* an efficient algorithm for this setting. However, it appears that that their regret is $O(n\sqrt{T})$ rather than $O(\sqrt{nT})$ — their regret bounds are stated in terms of diameters of the decision and cost spaces, but we can bound these in terms of $n$, which leads to the $O(n\sqrt{T})$ regret for their algorithm.

## 3.2 With Bandit Information

We now present the Geometric Hedge algorithm (shown in Algorithm 3.1) that achieves low expected regret for the setting where only the observed loss, $\ell_t = L_t \cdot x_t$, is received as feedback. This algorithm is motivated by the algorithms in Auer et al. [1998] (designed for the $K$-arm case), which use **Hedge** (with estimated losses) along with a $\gamma$ probability of exploration.

---

**Algorithm** GEOMETRICHEDGE$(D, \gamma, \eta)$
$\forall x \in D, p_1(x) \leftarrow \frac{1}{|D|}$
**for** $t \leftarrow 1$ **to** $T$
  $\forall x \in D, \widehat{p}_t(x) = (1 - \gamma)p_t(x) + \frac{\gamma}{n}\mathbf{1}\{x \in \text{spanner}\}$
  Sample $x_t$ according to distribution $\widehat{p}_t$
  Incur and observe loss $\ell_t := L_t^\dagger x_t$
  $C_t := \mathbb{E}_{\widehat{p}_t}[xx^\dagger]$
  $\widehat{L}_t := \ell_t C_t^{-1} x_t$
  $\forall x \in D, p_{t+1}(x) \propto p_t(x)e^{-\eta \widehat{L}_t^\dagger x}$

---

In the Geometric Hedge algorithm, there is a $\gamma$ probability of exploring with the spanner on each round (motivated by Awerbuch and Kleinberg [2004]). The estimated losses we feed into **Hedge** are determined by the estimator $\widehat{L}_t$ of $L_t$. Note that the algorithm is well defined as $C_t$ is always non-singular. The following lemma shows why this estimator is sensible.

**Lemma 3.2.** *On each round $t$, $\widehat{L}_t$ is an unbiased estimator for the true loss vector $L_t$.*

*Proof.* $\widehat{L}_t = \ell_t C_t^{-1} x_t = (L_t \cdot x_t)C_t^{-1} x_t = C_t^{-1} x_t(x_t^\dagger L_t)$. Therefore

$$\mathbb{E}[\widehat{L}_t] = \mathbb{E}[C_t^{-1} x_t(x_t^\dagger L_t)] = C_t^{-1}\mathbb{E}[x_t x_t^\dagger]L_t = C_t^{-1} C_t L_t = L_t$$

where all the expectations are over the random choice of $x_t$ drawn from $\widehat{p}_t$. $\qquad\qquad\square$

In the $K$-arm case, where $n = K$ and $D = \{\vec{e}_1, \ldots, \vec{e}_K\}$, Algorithm 3.1 specializes to the **Exp3** algorithm of Auer et al. [1998].

Note that if $|D|$ is exponential in the dimension $n$ then in general, maintaining and sampling from the distributions $p_t$ and $\widehat{p}_t$ is very expensive in terms of running time. However in many special cases of interest, this can actually be implemented efficiently.

We now state the main technical result of the paper.

**Theorem 3.3.** *Let $\gamma = \frac{n^{3/2}}{\sqrt{T}}$ and $\eta = \frac{1}{\sqrt{nT}}$ in Algorithm 3.1. For any sequence $L_1, \ldots, L_T$ of admissible loss vectors, let $R$ denote the regret of Algorithm 3.1 on this sequence. Then*

$$\mathbb{E}R \leq \ln|D|\sqrt{nT} + 2n^{3/2}\sqrt{T}$$

As before, since we may replace $D$ with a set of size $O((nT)^{n/2})$ for an additional regret of only $\sqrt{nT}$, the regret is $O^*(n^{3/2}\sqrt{T})$. Moreover, if $|D| \leq c^n$ for some constant $c$, as is the case for the online shortest path problem, then $\mathbb{E}R = O(n^{3/2}\sqrt{T})$.

## 3.3 Analysis of Algorithm 3.1

In this section, we prove Theorem 3.3. We start by providing the following bound on the sizes of the estimated loss vectors used by Algorithm 3.1.

**Lemma 3.4.** *For each $x \in D$ and $1 \leq t \leq T$, the estimated loss vector $\widehat{L}_t$ satisfies*

$$|\widehat{L}_t \cdot x| \leq \frac{n^2}{\gamma}$$

*Proof.* First, let us examine $C_t$. Let $\lambda_1, \ldots, \lambda_n$ be the eigenvalues of $C_t$, and $v_1, \ldots, v_n$ be the corresponding (orthonormal) eigenvectors. Since $C_t := \mathbb{E}_{\widehat{p}_t}\left[xx^\dagger\right]$ and $\lambda_i = v_i^\dagger C_t v_i$, we have

$$\lambda_i = v_i^\dagger \mathbb{E}_{\widehat{p}_t}\left[xx^\dagger\right]v_i = \sum_{x \in D} \widehat{p}_t(x)(x \cdot v_i)^2 \tag{1}$$

and so

$$\lambda_i = \sum_{x \in D} \widehat{p}_t(x)(x \cdot v_i)^2 \geq \sum_{x \in \text{spanner}} \widehat{p}_t(x)(x \cdot v_i)^2 \geq \sum_{j=1}^{n} \frac{\gamma}{n}(\vec{e}_j \cdot v_i)^2 = \frac{\gamma}{n}\|v_i\|^2 = \frac{\gamma}{n}$$

It follows that the eigenvalues $\lambda_1^{-1}, \ldots \lambda_n^{-1}$ of $C_t^{-1}$ are each at most $\frac{n}{\gamma}$.

Hence, for each $x$

$$|\widehat{L}_t \cdot x| = |\ell_t C_t^{-1} x_t \cdot x| \leq \frac{n}{\gamma}|\ell_t|\,\|x_t\|_2\|x\|_2 \leq \frac{n^2}{\gamma}$$

where we have used the upper bound on the eigenvalues and the upper bound of $\sqrt{n}$ for $x \in D$. $\quad\square$

The following proposition is Theorem 3.1 in Auer et al. [1998], restated in our notation (for losses instead of gains). We state it here without proof. Denote $\Phi_M(\eta) := \frac{e^{M\eta} - 1 - M\eta}{M^2}$.

**Proposition 3.5.** *(from Auer et al. [1998])For every $x^* \in D$, the sequence of estimated loss vectors $\widehat{L}_1, \ldots, \widehat{L}_T$ and the probability distributions $p_1, \ldots p_T$ satisfy*

$$\sum_{t=1}^{T}\sum_{x \in D} p_t(x)\widehat{L}_t \cdot x \leq \sum_{t=1}^{T} \widehat{L}_t \cdot x^* + \frac{\ln|D|}{\eta} + \frac{\Phi_M(\eta)}{\eta}\sum_{t=1}^{T}\sum_{x \in D} p_t(x)(\widehat{L}_t \cdot x)^2$$

*where $M = n^2/\gamma$ is an upper bound on $|\widehat{L} \cdot x|$.*

Before we are ready to complete the proof, two technical lemmas are useful.

**Lemma 3.6.** *For each $x \in D$ and $1 \leq t \leq T$,*

$$\mathbb{E}_{x_t \sim \widehat{p}_t}\left(\left(\widehat{L}_t \cdot x\right)^2\right) \leq x^\dagger C_t^{-1} x$$

*Proof.* Using that $\mathbb{E}\left(\left(\widehat{L}_t \cdot x\right)^2\right) = x^\dagger \mathbb{E}\left(\widehat{L}_t \widehat{L}_t^\dagger\right)x$, we have

$$x^\dagger \mathbb{E}\left(\widehat{L}_t \widehat{L}_t^\dagger\right)x = x^\dagger \mathbb{E}\left(\ell_t^2 C_t^{-1} x_t x_t^\dagger C_t^{-1}\right)x \leq x^\dagger C_t^{-1} \mathbb{E}\left(x_t x_t^\dagger\right)C_t^{-1} x = x^\dagger C_t^{-1} x \quad\quad\square$$

**Lemma 3.7.** *For each $1 \leq t \leq T$,*

$$\sum_{x \in D} \widehat{p}_t(x) x^\dagger C_t^{-1} x = n$$

*Proof.* The singular value decomposition of $C_t^{-1}$ is $VBV^\dagger$ where $B$ is diagonal (with the inverse eigenvalues as the diagonal entries) and $V$ is orthogonal (with the columns being the eigenvectors). This implies that $x^\dagger C_t^{-1} x = \sum_i \lambda_i^{-1}(x \cdot v_i)^2$. Using Equation 1, it follows that

$$\sum_{x \in D} \widehat{p}_t(x) x^\dagger C_t^{-1} x = \sum_{x \in D} \widehat{p}_t(x) \sum_{i=1}^{n} \lambda_i^{-1}(x \cdot v_i)^2 = \sum_{i=1}^{n} \lambda_i^{-1}\sum_{x \in D} \widehat{p}_t(x)(x \cdot v_i)^2 = \sum_{i=1}^{n} 1 = n \quad\square$$

We are now ready to complete the proof of Theorem 3.3.

*Proof.* We now have, for any $x^* \in D$,

$$\sum_{t,x} \widehat{p}_t(x)\widehat{L}_t \cdot x = \sum_{t=1}^{T}\sum_{x \in D}\left(\left(1 - \gamma\right)p_t(x) + \frac{\gamma}{n}\mathbf{1}\{\exists j : x = \vec{e}_j\}\right)\widehat{L}_t \cdot x$$

$$\leq (1 - \gamma)\left(\sum_{t=1}^{T} \widehat{L}_t \cdot x^* + \frac{\ln|D|}{\eta} + \frac{\Phi_M(\eta)}{\eta}\sum_{t,x} p_t(x)(\widehat{L}_t \cdot x)^2\right) + \sum_{t=1}^{T}\sum_{j=1}^{n} \frac{\gamma}{n}\widehat{L}_t \cdot \vec{e}_j$$

$$\leq \sum_{t=1}^{T} \widehat{L}_t \cdot x^* + \frac{\ln|D|}{\eta} + \frac{\Phi_M(\eta)}{\eta}\sum_{t,x} \widehat{p}_t(x)(\widehat{L}_t \cdot x)^2 + \sum_{t=1}^{T}\sum_{j=1}^{n} \frac{\gamma}{n}\widehat{L}_t \cdot \vec{e}_j$$

where the last step uses $(1-\gamma)p_t(x) \le \widehat{p}_t(x)$. Taking expectations and using the unbiased property,

$$\mathbb{E}\left[\sum_{t,x} \widehat{p}_t(x)\widehat{L}_t \cdot x\right] = \sum_{t=1}^{T} L_t \cdot x^* + \frac{\ln|D|}{\eta} + \frac{\Phi_M(\eta)}{\eta}\mathbb{E}\left[\sum_{t,x}\widehat{p}_t(x)(\widehat{L}_t \cdot x)^2\right] + \sum_{t=1}^{T}\sum_{j=1}^{n}\frac{\gamma}{n}L_t \cdot \vec{e}_j$$

$$\le \sum_{t=1}^{T} L_t \cdot x^* + \frac{\ln|D|}{\eta} + \frac{\Phi_M(\eta)}{\eta}\mathbb{E}\left[\sum_{t,x}\widehat{p}_t(x)\underset{x_t\sim\widehat{p}_t}{\mathbb{E}}(\widehat{L}_t \cdot x)^2\right] + \gamma T$$

$$\le \sum_{t=1}^{T} L_t \cdot x^* + \frac{\ln|D|}{\eta} + \frac{\Phi_M(\eta)}{\eta}nT + \gamma T$$

where we have used Lemmas 3.6 and 3.7 in the last step.

Setting $\gamma = \frac{n^{3/2}}{\sqrt{T}}$ and $\eta = \frac{1}{\sqrt{nT}}$ gives $M\eta = n^2\eta/\gamma \le 1$, which implies that

$$\Phi_M(\eta) = \frac{e^{M\eta} - 1 - M\eta}{M^2} \le \frac{M^2\eta^2}{M^2} = \eta^2$$

where the inequality comes from that for $\alpha \le 1$, $e^\alpha \le 1 + \alpha + \alpha^2$. With the above, we have

$$\mathbb{E}\left[\sum_{t,x}\widehat{p}_t(x)\widehat{L}_t \cdot x\right] \le \sum_{t=1}^{T} L_t \cdot x^* + \ln|D|\sqrt{nT} + 2n^{3/2}\sqrt{T}$$

The proof is completed by noting that

$$\mathbb{E}\left[\sum_{t,x}\widehat{p}_t(x)\widehat{L}_t \cdot x\right] = \mathbb{E}\left[\sum_{t,x}\widehat{p}_t(x)\mathbb{E}\left(\widehat{L}_t \mid \mathcal{H}_t\right) \cdot x\right] = \mathbb{E}\left[\sum_{t,x}\widehat{p}_t(x)L_t \cdot x\right] = \mathbb{E}\left(\sum_t L_t \cdot x_t\right)$$

is the expected total loss of the algorithm. $\qquad\square$

# 4 Lower Bounds

## 4.1 With Full Information

We now present a family of distributions which establishes an $\Omega(\sqrt{nT})$ lower bound for i.i.d. loss vectors in the full information setting. In the remainder of the paper, we assume for convenience that the incurred losses are in the interval $[-1, 1]$ rather than $[0, 1]$. (This changes the bounds by at most a factor of 2.)

**Example 4.1.** For a given $S \subseteq \{1, \ldots, n\}$ and $0 < \varepsilon < 1$, we define a random loss vector $L$ as follows. Choose $i \in \{1, \ldots, n\}$ uniformly at random. Let $\sigma \in \pm 1$ be 1 with probability $(1 + \varepsilon)/2$ and $-1$ otherwise. Set

$$L = \begin{cases} \sigma\vec{e}_i & \text{if } i \in S \\ -\sigma\vec{e}_i & \text{if } i \notin S \end{cases}$$

Let $\mathcal{D}_{S,\varepsilon}$ denote the distribution of $L$.

**Theorem 4.2.** *Suppose the decision set $D$ is the unit hypercube $\{-1, 1\}^n$. For any full-information linear optimization algorithm $\mathcal{A}$, and for any positive integer $T$, there exists $S \subseteq \{1, \ldots, n\}$ such that for loss vectors $L_1, \ldots, L_T$ sampled i.i.d. according to $\mathcal{D}_{S,\sqrt{n/T}}$, the expected regret is $\Omega(\sqrt{nT})$.*

*Proof sketch.* Clearly, for each $S$ and $\varepsilon$, the optimal decision vector for loss vectors sampled i.i.d. according to $\mathcal{D}_{S,\varepsilon}$ is the vector $(x_1, \ldots, x_n)$ where $x_i = -1$ if $i \in S$ and 1 otherwise.

Suppose $S$ is chosen uniformly at random. In this case, it is clear that the optimal algorithm chooses decision $(x_1, \ldots, x_n)$ where for each $i$, the sign of $x_i$ is the same as the minority of past occurrences of loss vectors $\pm e_i$ (in case of a tie, the value of $x_i$ doesn't matter).

Note that at every time step when the empirical minority incorrectly predicts the bias for coordinate $i$, the optimal algorithm incurs expected regret $\Omega(\varepsilon/n)$. By a standard application of Stirling's

estimates, one can show that until coordinate $i$ has been chosen $\Omega(1/\varepsilon^2)$ times, the probability that the empirical majority disagrees with the long-run average is $\Omega(1)$. In expectation, this requires $\Omega(n/\varepsilon^2)$ time steps. Summing over the $n$ arms, the overall expected regret is thus at least $\Omega(n(\varepsilon/n)\min\{T, n/\varepsilon^2\} = \Omega(\min\{\varepsilon T, n/\varepsilon\})$. Setting $\varepsilon = \sqrt{n/T}$ yields the desired bound. $\qquad\square$

## 4.2  With Bandit Information

Next we prove that the same decision set $\{0,1\}^n$ and family of distributions $\mathcal{D}_{S,\varepsilon}$ can be used to establish an $\Omega(n\sqrt{T})$ lower bound in the bandit setting.

**Theorem 4.3.** *Suppose the decision set $D$ is the unit hypercube $\{0,1\}^n$. For any bandit linear optimization algorithm $\mathcal{A}$, and for any positive integer $T$, there exists $S \subseteq \{1,\ldots,n\}$ such that for loss functions $L_1,\ldots,L_T$ sampled i.i.d. according to $\mathcal{D}_{S,n/\sqrt{T}}$, the expected regret is $\Omega(n\sqrt{T})$.*

*Proof sketch.* Again, for each $S$ and $\varepsilon$, the optimal decision vector for loss vectors sampled i.i.d. according to $\mathcal{D}_{S,\varepsilon}$ is just the indicator vector for the set $S$.

Suppose $S$ is chosen uniformly at random. Unlike the proof of Theorem 4.2, we do not attempt to characterize the optimal algorithm for this setting.

Note that, for every $1 \le i \le n$, every time step when the algorithm incorrectly sets $x_i \ne \mathbf{1}\{i \in S\}$, contributes $\Omega(\varepsilon/n)$ to the expected regret. Let us fix $i \in \{1,\ldots,n\}$ and prove a lower bound on its expected contribution to the total regret. To simplify matters, let us consider the best algorithm conditioned on the value of $S \setminus \{i\}$. It is not hard to see that the problem of guessing the membership of $i$ in $S$ based on $t$ past measurements can be recast as a problem of deciding between two possible means which differ by $\varepsilon/n$, given a sequence of $t$ i.i.d. Bernoulli random variables with one of the unknown mean, where each of the means is *a priori* equally likely. But for this problem, the error probability is $\Omega(1)$ unless $t = \Omega((n/\varepsilon)^2)$. Thus we have shown that the expected contribution of coordinate $i$ to the total regret is $\Omega(\min\{T, (n/\varepsilon)^2\}\varepsilon/n)$. Summing over the $n$ arms gives an overall expected regret of $\Omega(\min\{\varepsilon T, n^2/\varepsilon\}$. Setting $\varepsilon = n/\sqrt{T}$ completes the proof. $\qquad\square$

## References

P. Auer, N. Cesa-Bianchi, Y. Freund, and R. E. Schapire. Gambling in a rigged casino: the adversarial multi-armed bandit problem. In *Proceedings of the 36th Annual Symposium on Foundations of Computer Science (1995)*. IEEE Computer Society Press, Los Alamitos, CA, extended version, 24pp., dated June 8, 1998. Available from R. Schapire's website.

B. Awerbuch and R. Kleinberg. Adaptive routing with end-to-end feedback: Distributed learning and geometric approaches. In *Proceedings of the 36th ACM Symposium on Theory of Computing (STOC)*, 2004.

P. Bartlett, V. Dani, T. P. Hayes, S. M. Kakade, A. Rakhlin, and A. Tewari. High probability regret bounds for online optimization (working title). Manuscript, 2007.

V. Dani, T. P. Hayes, and S. M. Kakade. Stochastic linear optimization under bandit feedback. In submission, 2008.

Varsha Dani and Thomas P. Hayes. Robbing the bandit: Less regret in online geometric optimization against an adaptive adversary. In *Proceedings of the 17th ACM-SIAM Symposium on Discrete Algorithms (SODA)*, 2006.

Y. Freund and R. Schapire. A decision-theoretic generalization of on-line learning and an application to boosting. *Journal of Computer and System Sciences*, 55(1):119–139, 1997.

A. Gyorgy, T. Linder, G. Lugosi, and G. Ottucsak. The on-line shortest path problem under partial monitoring. *Journal of Machine Learning Research*, 8:2369–2403, 2007.

Adam Kalai and Santosh Vempala. Efficient algorithms for online decision problems. *J. Comput. Syst. Sci.*, 71 (3):291–307, 2005. ISSN 0022-0000. doi: http://dx.doi.org/10.1016/j.jcss.2004.10.016.

T. L. Lai and H. Robbins. Asymptotically efficient adaptive allocation rules. *Advances in Applied Mathematics*, 6:4–25, 1985.

H.B. McMahan and A. Blum. Online geometric optimization in the bandit setting against an adaptive adversary. In *Proceedings of the 17th Annual Conference on Learning Theory (COLT)*, 2004.

H. Robbins. Some aspects of the sequential design of experiments. *Bulletin of the American Mathematical Society*, 55:527–535, 1952.

Eiji Takimoto and Manfred K. Warmuth. Path kernels and multiplicative updates. *J. Mach. Learn. Res.*, 4: 773–818, 2003. ISSN 1533-7928.

